# Analytical Mean Squared Error Curves in Temporal Difference Learning

**Satinder Singh**
Department of Computer Science
University of Colorado
Boulder, CO 80309-0430
baveja@cs.colorado.edu

**Peter Dayan**
Brain and Cognitive Sciences
E25-210, MIT
Cambridge, MA 02139
bertsekas@lids.mit.edu

## Abstract

We have calculated analytical expressions for how the bias and variance of the estimators provided by various temporal difference value estimation algorithms change with offline updates over trials in absorbing Markov chains using lookup table representations. We illustrate classes of learning curve behavior in various chains, and show the manner in which TD is sensitive to the choice of its step-size and eligibility trace parameters.

## 1 INTRODUCTION

A reassuring theory of asymptotic convergence is available for many reinforcement learning (RL) algorithms. What is not available, however, is a theory that explains the finite-term learning curve behavior of RL algorithms, e.g., what are the different kinds of learning curves, what are their key determinants, and how do different problem parameters effect rate of convergence. Answering these questions is crucial not only for making useful comparisons between algorithms, but also for developing hybrid and new RL methods. In this paper we provide preliminary answers to some of the above questions for the case of absorbing Markov chains, where mean square error between the estimated and true predictions is used as the quantity of interest in learning curves.

Our main contribution is in deriving the analytical update equations for the two components of MSE, bias and variance, for popular Monte Carlo (MC) and TD($\lambda$) (Sutton, 1988) algorithms. These derivations are presented in a larger paper. Here we apply our theoretical results to produce analytical learning curves for TD on two specific Markov chains chosen to highlight the effect of various problem and algorithm parameters, in particular the definite trade-offs between step-size, $\alpha$, and eligibility-trace parameter, $\lambda$. Although these results are for specific problems, we

believe that many of the conclusions are intuitive or have previous empirical support, and may be more generally applicable.

## 2 ANALYTICAL RESULTS

A random walk, or trial, in an absorbing Markov chain with only terminal payoffs produces a sequence of states terminated by a payoff. The prediction task is to determine the expected payoff as a function of the start state $i$, called the optimal value function, and denoted $\mathbf{v}^*$. Accordingly, $v_i^* = E\{r|s_1 = i\}$, where $s_t$ is the state at step $t$, and $r$ is the random terminal payoff. The algorithms analysed are iterative and produce a sequence of estimates of $\mathbf{v}^*$ by repeatedly combining the result from a new trial with the old estimate to produce a new estimate. They have the form: $v_i(t) = v_i(t-1) + \alpha(t)\delta_i(t)$ where $\mathbf{v}(t) = \{v_i(t)\}$ is the estimate of the optimal value function after $t$ trials, $\delta_i(t)$ is the result for state $i$ based on random trial $t$, and the step-size $\alpha(t)$ determines how the old estimate and the new result are combined. The algorithms differ in the $\delta$s produced from a trial.

Monte Carlo algorithms use the final payoff that results from a trial to define the $\delta_i(t)$ (e.g., Barto & Duff, 1994). Therefore in MC algorithms the estimated value of a state is unaffected by the estimated value of any other state. The main contribution of TD algorithms (Sutton, 1988) over MC algorithms is that they update the value of a state based not only on the terminal payoff but also on the the estimated values of the intervening states. When a state is first visited, it initiates a short-term memory process, an eligibility trace, which then decays exponentially over time with parameter $\lambda$. The amount by which the value of an intervening state combines with the old estimate is determined in part by the magnitude of the eligibility trace at that point.

In general, the initial estimate $\mathbf{v}(0)$ could be a random vector drawn from some distribution, but often $\mathbf{v}(0)$ is fixed to some initial value such as zero. In either case, subsequent estimates, $\mathbf{v}(t); t > 0$, will be random vectors because of the random $\delta$s. The random vector $\mathbf{v}(t)$ has a bias vector $\mathbf{b}(t) \stackrel{def}{=} E\{\mathbf{v}(t) - \mathbf{v}^*\}$ and a covariance matrix $C(t) \stackrel{def}{=} E\{(\mathbf{v}(t) - E\{\mathbf{v}(t)\})(\mathbf{v}(t) - E\{\mathbf{v}(t)\})^T\}$. The scalar quantity of interest for learning curves is the weighted MSE as a function of trial number $t$, and is defined as follows:

$$\text{MSE}(t) = \sum_i p_i(E\{(v_i(t) - v_i^*)^2\}) = \sum_i p_i(b_i^2(t) + C_{ii}(t)),$$

where $p_i = (\mu^T[I-Q]^{-1})_i / \sum_j (\mu^T[I-Q]^{-1})_j$ is the weight for state $i$, which is the expected number of visits to $i$ in a trial divided by the expected length of a trial[1] ($\mu_i$ is the probability of starting in state $i$; $Q$ is the transition matrix of the chain).

In this paper we present results just for the standard TD($\lambda$) algorithm (Sutton, 1988), but we have analysed (Singh & Dayan, 1996) various other TD-like algorithms (e.g., Singh & Sutton, 1996) and comment on their behavior in the conclusions. Our analytical results are based on two non-trivial assumptions: first that lookup tables are used, and second that the algorithm parameters $\alpha$ and $\lambda$ are functions of the trial number alone rather than also depending on the state. We also make two assumptions that we believe would not change the general nature of the results obtained here: that the estimated values are updated offline (after the end of each trial), and that the only non-zero payoffs are on the transitions to the terminal states. With the above caveats, our analytical results allow rapid computation of *exact* mean square error (MSE) learning curves as a function of trial number.

## 2.1   BIAS, VARIANCE, And MSE UPDATE EQUATIONS

The analytical update equations for the bias, variance and MSE are complex and their details are in Singh & Dayan (1996) — they take the following form in outline:

$$\mathbf{b}(t) = \boldsymbol{a}^m + B^m \mathbf{b}(t-1) \tag{1}$$

$$C(t) = A^S + B^S C(t-1) + f^S(\mathbf{b}(t-1)) \tag{2}$$

where matrix $B^m$ depends linearly on $\alpha(t)$ and $B^S$ and $f^S$ depend at most quadratically on $\alpha(t)$. We coded this detail in the C programming language to develop a software tool[2] whose rapid computation of exact MSE error curves allowed us to experiment with many different algorithm and problem parameters on many Markov chains. Of course, one could have averaged together many empirical MSE curves obtained via simulation of these Markov chains to get approximations to the analytical MSE error curves, but in many cases MSE curves that take minutes to compute analytically take days to derive empirically on the same computer for five significant digit accuracy. Empirical simulation is particularly slow in cases where the variance converges to non-zero values (because of constant step-sizes) with long tails in the asymptotic distribution of estimated values (we present an example in Figure 1c). Our analytical method, on the other hand, computes exact MSE curves for $L$ trials in $O(|state\ space|^3 L)$ steps regardless of the behavior of the variance and bias curves.

## 2.2   ANALYTICAL METHODS

Two consequences of having the analytical forms of the equations for the update of the mean and variance are that it is possible to optimize schedules for setting $\alpha$ and $\lambda$ and, for fixed $\lambda$ and $\alpha$, work out terminal rates of convergence for $\mathbf{b}$ and $C$.

**Computing one-step optimal $\alpha$'s:** Given a particular $\lambda$, the effect on the MSE of a single step for any of the algorithms is quadratic in $\alpha$. It is therefore straightforward to calculate the value of $\alpha$ that minimises $MSE(t)$ at the next time step. This is called the *greedy* value of $\alpha$. It is not clear that if one were interested in minimising $MSE(t + t')$, one would choose successive $\alpha(u)$ that greedily minimise $MSE(t); MSE(t+1); \ldots$. In general, one could use our formulæ and dynamic programming to optimise a whole schedule for $\alpha(u)$, but this is computationally challenging.

Note that this technique for setting greedy $\alpha$ assumes complete knowledge about the Markov chain and the initial bias and covariance of $\mathbf{v}(0)$, and is therefore not directly applicable to realistic applications of reinforcement learning. Nevertheless, it is a good analysis tool to approximate omniscient optimal step-size schedules, eliminating the effect of the choice of $\alpha$ when studying the effect of the $\lambda$.

**Computing one-step optimal $\lambda$'s:** Calculating analytically the $\lambda$ that would minimize $MSE(t)$ given the bias and variance at trial $t - 1$ is substantially harder because terms such as $[I - \lambda(t)Q]^{-1}$ appear in the expressions. However, since it is possible to compute $MSE(t)$ for any choice of $\lambda$, it is straightforward to find to any desired accuracy the $\lambda_g(t)$ that gives the lowest resulting $MSE(t)$. This is possible only because $MSE(t)$ can be computed very cheaply using our analytical equations.

The caveats about greediness in choosing $\alpha_g(t)$ also apply to $\lambda_g(t)$. For one of the Markov chains, we used a stochastic gradient ascent method to optimise $\lambda(u)$ and

[2]The analytical MSE error curve software is available via anonymous ftp from the following address: ftp.cs.colorado.edu /users/baveja/AMse.tar.Z

$\alpha(u)$ to minimise MSE$(t + t')$ and found that it was not optimal to choose $\lambda_g(t)$ and $\alpha_g(t)$ at the first step.

**Computing terminal rates of convergence:** In the update equations 1 and 2, $\mathbf{b}(t)$ depends linearly on $\mathbf{b}(t-1)$ through a matrix $B^m$; and $C(t)$ depends linearly on $C(t-1)$ through a matrix $B^S$. For the case of fixed $\alpha$ and $\lambda$, the maximal and minimal eigenvalues of $B^m$ and $B^S$ determine the fact and speed of convergence of the algorithms to finite endpoints. If the modulus of the real part of any of the eigenvalues is greater than 1, then the algorithms will not converge in general. We observed that the mean update is more stable than the mean square update, i.e., appropriate eigenvalues are obtained for larger values of $\alpha$ (we call the largest feasible $\alpha$ the largest learning rate for which TD will converge). Further, we know that the mean converges to $\mathbf{v}^*$ if $\alpha$ is sufficiently small that it converges at all, and so we can determine the terminal covariance. Just like the delta rule, these algorithms converge at best to an $\epsilon$-ball for a constant finite step-size. This amounts to the MSE converging to a fixed value, which our equations also predict. Further, by calculating the eigenvalues of $B^m$, we can calculate an estimate of the rate of decrease of the bias.

## 3  LEARNING CURVES ON SPECIFIC MARKOV CHAINS

We applied our software to two problems: a symmetric random walk (SRW), and a Markov chain for which we can control the frequency of returns to each state in a single run (we call this the *cyclicity* of the chain).

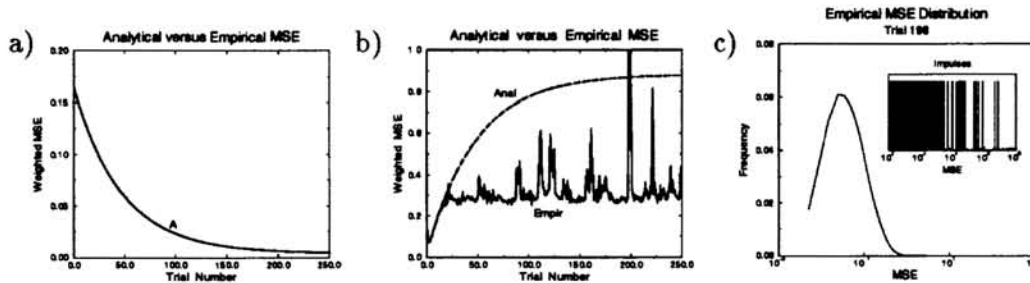

Figure 1: Comparing Analytical and Empirical MSE curves. a) analytical and empirical learning curves obtained on the 19 state SRW problem with parameters $\alpha = 0.01$, $\lambda = 0.9$. The empirical curve was obtained by averaging together more than three million simulation runs, and the analytical and empirical MSE curves agree up to the fourth decimal place; b) a case where the empirical method fails to match the analytical learning curve after more than 15 million runs on a 5 state SRW problem. The empirical learning curve is very spiky. c) Empirical distribution plot over 15.5 million runs for the MSE at trial 198. The inset shows impulses at actual sample values greater than 100. The largest value is greater than 200000.

**Agreement:** First, we present empirical confirmation of our analytical equations on the 19 state SRW problem. We ran TD$(\lambda)$ for specific choices of $\alpha$ and $\lambda$ for more than three million simulation runs and averaged the resulting empirical weighted MSE error curves. Figure 1a shows the analytical and empirical learning curves, which agree to within four decimal places.

**Long-Tails of Empirical MSE distribution:** There are cases in which the agreement is apparently much worse (see Figure 1b). This is because of the surprisingly *long tails* for the empirical MSE distribution – Figure 1c shows an example for a 5

state SRW. This points to interesting structure that our analysis is unable to reveal.

**Effect of $\alpha$ and $\lambda$:** Extensive studies on the 19 state SRW that we do not have space to describe fully show that: H1) for each algorithm, increasing $\alpha$ while holding $\lambda$ fixed increases the asymptotic value of MSE, and similarly for increasing $\lambda$ whilst holding $\alpha$ constant; H2) larger values of $\alpha$ or $\lambda$ (except $\lambda$ very close to 1) lead to faster convergence to the asymptotic value of MSE if there exists one; H3) in general, for each algorithm as one decreases $\lambda$ the reasonable range of $\alpha$ shrinks, i.e., larger $\alpha$ can be used with larger $\lambda$ without causing excessive MSE. The effect in H3 is counter-intuitive because larger $\lambda$ tends to amplify the effective step-size and so one would expect the opposite effect. Indeed, this increase in the range of feasible $\alpha$ is not strictly true, especially very near $\lambda = 1$, but it does seem to hold for a large range of $\lambda$.

**MC versus TD($\lambda$):** Sutton (1988) and others have investigated the effect of $\lambda$ on the empirical MSE at small trial numbers and consistently shown that TD is better for some $\lambda < 1$ than MC ($\lambda = 1$). Figure 2a shows substantial changes as a function of trial number in the value of $\lambda$ that leads to the lowest MSE. This effect is consistent with hypotheses $H1$-$H3$. Figure 2b confirms that this remains true even if greedy choices of $\alpha$ tailored for each value of $\lambda$ are used. Curves for different values of $\lambda$ yield minimum MSE over different trial number segments. We observed these effects on several Markov chains.

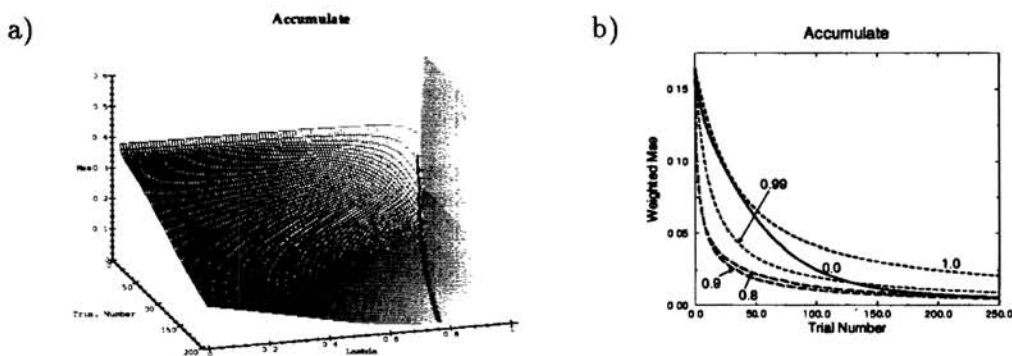

Figure 2: U-shaped Curves. a) Weighted MSE as a function of $\lambda$ and trial number for fixed $\alpha = 0.05$ (minimum in $\lambda$ shown as a black line). This is a 3-d version of the U-shaped curves in Sutton (1988), with trial number being the extra axis. b) Weighted MSE as a function of trial number for various $\lambda$ using greedy $\alpha$. Curves for different values of $\lambda$ yield minimum MSE over different trial number segments.

**Initial bias:** Watkins (1989) suggested that $\lambda$ trades off bias for variance, since $\lambda \sim 1$ has low bias, but potentially high variance, and conversely for $\lambda \sim 0$. Figure 3a confirms this in a problem which is a little like a random walk, except that it is highly cyclic so that it returns to each state many times in a single trial. If the initial bias is high (low), then the initial greedy value of $\lambda$ is high (low). We had expected the asymptotic greedy value of $\lambda$ to be 0, since once $b(t) \sim 0$, then $\lambda = 0$ leads to lower variance updates. However, Figure 3a shows a non-zero asymptote – presumably because larger learning rates can be used for $\lambda > 0$, because of covariance. Figure 3b shows, however, that there is little advantage in choosing $\lambda$ cleverly except in the first few trials, at least if good values of $\alpha$ are available.

**Eigenvalue stability analysis:** We analysed the eigenvalues of the covariance update matrix $B^S$ (c.f. Equation 2) to determine maximal fixed $\alpha$ as a function of $\lambda$. Note that larger $\alpha$ tends to lead to faster learning, provided that the values converge. Figure 4a shows the largest eigenvalue of $B^S$ as a function of $\lambda$ for various

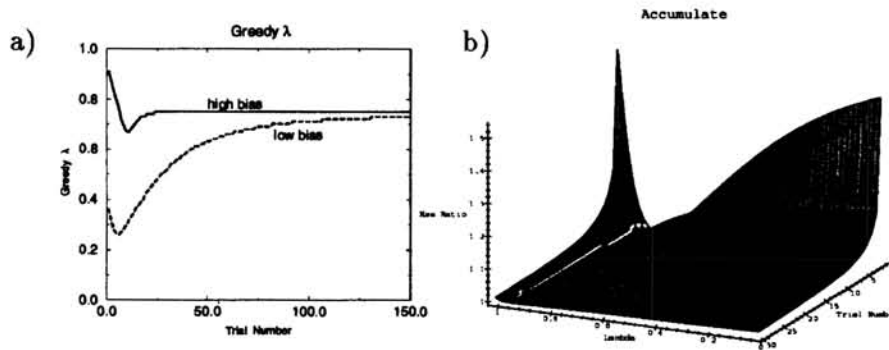

Figure 3: Greedy $\lambda$ for a highly cyclic problem. a) Greedy $\lambda$ for high and low initial bias (using greedy $\alpha$). b) Ratio of MSE for given value of $\lambda$ to that for greedy $\lambda$ at each trial. The greedy $\lambda$ is used for every step.

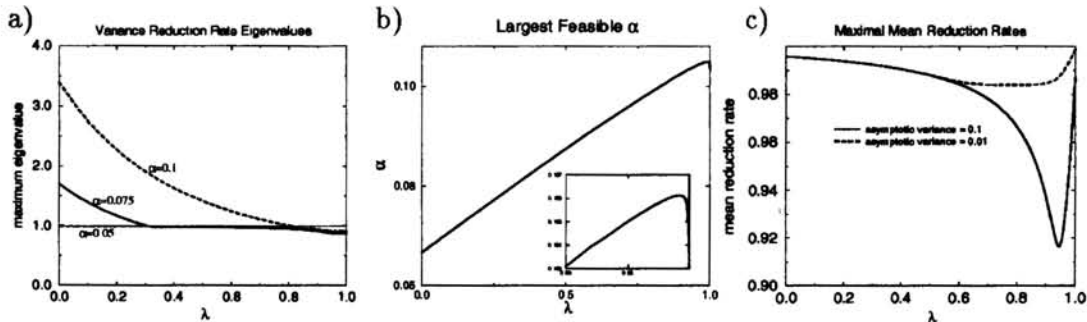

Figure 4: Eigenvalue analysis of covariance reduction. a) Maximal modulus of the eigenvalues of $B^S$. These determine the rate of convergence of the variance. Values greater than 1 lead to instability. b) Largest $\alpha$ such that the covariance is bounded. The inset shows a blowup for $0.9 \leq \lambda \leq 1$. Note that $\lambda = 1$ is not optimal. c) Maximal bias reduction rates as a function of $\lambda$, after controlling for asymptotic variance (to 0.1 and 0.01) by choosing appropriate $\alpha$'s. Again, $\lambda < 1$ is optimal.

$\alpha$. If this eigenvalue is larger than 1, then the algorithm will diverge – a behavior that we observed in our simulations. The effect of hypothesis H3 above is evident – for larger $\lambda$, only smaller $\alpha$ can be used. Figure 4b shows this in more graphic form, indicating the largest $\alpha$ that leads to stable eigenvalues for $B^S$. Note the reversal very close to $\lambda = 1$, which provides more evidence against the pure MC algorithm. The choice of $\alpha$ and $\lambda$ control both rate of convergence and the asymptotic MSE. In Figure 4c we control for the asymptotic variance by choosing appropriate $\alpha$s as a function of $\lambda$ and plot maximal eigenvalues of $B^m$ (c.f. Equation 1; it controls the terminal rate of convergence of the bias to zero) as a function of $\lambda$. Again, we see evidence for $TD$ over $MC$.

## 4 CONCLUSIONS

We have provided analytical expressions for calculating how the bias and variance of various TD and Monte Carlo algorithms change over iterations. The expressions themselves seem not to be very revealing, but we have provided many illustrations of their behavior in some example Markov chains. We have also used the analysis to calculate one-step optimal values of the step-size $\alpha$ and eligibility trace $\lambda$ parameters. Further, we have calculated terminal mean square errors and maximal bias reduction rates. Since all these results depend on the precise Markov chains chosen,

it is hard to make generalisations.

We have posited four general conjectures: H1) for constant $\lambda$, the larger $\alpha$, the larger the terminal MSE; H2) the larger $\alpha$ or $\lambda$ (except for $\lambda$ very close to 1), the faster the convergence to the asymptotic MSE, provided that this is finite; H3) the smaller $\lambda$, the *smaller* the range of $\alpha$ for which the terminal MSE is not excessive; H4) higher values of $\lambda$ are good for cases with high initial biases. The third of these is somewhat surprising, because the effective value of the step-size is really $\alpha/(1 - \lambda)$. However, the lower $\lambda$, the more the value of a state is based on the value estimates for nearby states. We conjecture that with small $\lambda$, large $\alpha$ can quickly lead to high correlation in the value estimates of nearby states and result in runaway variance updates.

Two main lines of evidence suggest that using values of $\lambda$ other than 1 (i.e., using a temporal difference rather than a Monte-Carlo algorithm) can be beneficial. First, the *greedy* value of $\lambda$ chosen to minimise the MSE at the end of the step (whilst using the associated greedy $\alpha$) remains away from 1 (see Figure 3). Second, the eigenvalue analysis of $B^S$ showed that the largest value of $\alpha$ that can be used is higher for $\lambda < 1$ (also the asymptotic speed with which the bias can be guaranteed to decrease fastest is higher for $\lambda < 1$).

Although in this paper we have only discussed results for the standard TD($\lambda$) algorithm (called Accumulate), we have also analysed Replace TD($\lambda$) of Singh & Sutton (1996) and various others. This analysis clearly provides only an early step to understanding the course of learning for TD algorithms, and has focused exclusively on prediction rather than control. The analytical expressions for MSE might lend themselves to general conclusions over whole classes of Markov chains, and our graphs also point to interesting unexplained phenomena, such as the apparent long tails in Figure 1c and the convergence of greedy values of $\lambda$ in Figure 3. Stronger analyses such as those providing large deviation rates would be desirable.

## Footnotes

[1]Other reasonable choices for the weights, $p_i$, would not change the nature of the results presented here.

## References

Barto, A.G. & Duff, M. (1994). Monte Carlo matrix inversion and reinforcement learning. NIPS 6, pp 687-694.

Singh, S.P. & Dayan, P. (1996). Analytical mean squared error curves in temporal difference learning. *Machine Learning*, submitted.

Singh, S.P. & Sutton, R.S. (1996). Reinforcement learning with replacing eligibility traces. *Machine Learning*, to appear.

Sutton, R.S. (1988). Learning to predict by the methods of temporal difference. *Machine Learning*, **3**, pp 9-44.

Watkins, C.J.C.H. (1989). *Learning from Delayed Rewards*. PhD Thesis. University of Cambridge, England.
